# Natural Actor-Critic for Road Traffic Optimisation

**Silvia Richter**
Albert-Ludwigs-Universität
Freiburg, Germany
si.richter@web.de

**Douglas Aberdeen**
National ICT Australia
Canberra, Australia
doug.aberdeen@anu.edu.au

**Jin Yu**
National ICT Australia
Canberra, Australia.
jin.yu@anu.edu.au

## Abstract

Current road-traffic optimisation practice around the world is a combination of hand tuned policies with a small degree of automatic adaption. Even state-of-the-art research controllers need good models of the road traffic, which cannot be obtained directly from existing sensors. We use a policy-gradient reinforcement learning approach to directly optimise the traffic signals, mapping currently deployed sensor observations to control signals. Our trained controllers are (theoretically) compatible with the traffic system used in Sydney and many other cities around the world. We apply two policy-gradient methods: (1) the recent natural actor-critic algorithm, and (2) a vanilla policy-gradient algorithm for comparison. Along the way we extend natural-actor critic approaches to work for distributed and *online* infinite-horizon problems.

## 1 Introduction

Optimising the performance of existing road networks is a cheap way to reduce the environmental, social, and financial impact of ever increasing volumes of traffic. Road traffic optimisation can be naturally cast as a reinforcement learning (RL) problem. Unfortunately it is in the hardest class of RL problems, having a continuous state space and infinite horizon, and being partially observable and difficult to model. We focus on the use of the natural actor-critic (NAC) algorithm [1] to solve this problem through online interaction with the traffic system.

The NAC algorithm is an elegant combination of 4 RL methods: (1) policy-gradient *actors* to allow local convergence under function approximation and partial observability; (2) *natural* gradients to incorporate curvature statistics into the gradient-ascent; (3) temporal-difference *critics* to reduce the variance of gradient estimates; (4) least-squares temporal difference methods to avoid wasting information from costly environment interactions.

One contribution of this work is an efficient online version of NAC that avoids large gradient steps, which cannot be guaranteed to be an improvement in the presence of stochastic gradients [2]. We compare this online NAC to a simple online policy-gradient (PG) approach, demonstrating that NAC converges in orders of magnitude less environment interactions. We also compare wall-clock convergence time, and suggest that environments which can be simulated quickly and accurately can be optimised faster with simple PG approaches rather than NAC.

This work has grown out of an interaction with the Sydney Road Traffic Authority. Our choice of controls, observations, and algorithms all aim for practical large-scale traffic control. Although our results are based on a simplified traffic simulation system, we could theoretically attach our learning system into real-world traffic networks. Our simple simulator results demonstrate better performance than the automatic adaption schemes used in current proprietary systems.

## 2 Traffic Control

The optimisation problem consists of finding signalling schedules for all intersections in the system that minimise the average travel time, or similar objectives. This is complicated by the fact that many of the influencing state variables cannot be readily measured. Most signal controllers in use today rely only on state information gained from inductive loops in the streets.

A *stream* is a sequence of cars making the same turn (or going straight) through an intersection. A *phase* is an interval where a subset of the lights at an intersection are green such that a set of streams that will not collide have right of way. A *signal cycle* is completed when each phase has been on once, the *cycle time* being the sum of phase times. Traditionally, control algorithms optimise traffic flow via the *phase scheme*, the *split*, and the *offset*. The *phase scheme* groups the signal lights into phases and determines their order. A *split* gives a distribution of the cycle time to the individual phases. *Offsets* can be introduced to coordinate neighbouring intersections, creating a "green wave" for the vehicles travelling along a main road.

Approaches to Traffic Control can be grouped into three categories. *Fixed time* control strategies are calculated off-line, based on historical data. TRANSYT, for example, uses evolutionary algorithms and hill-climbing optimisation [3]. *Traffic responsive* strategies are real-time, calculating their policies from car counts determined from inductive-loop detectors. SCOOT and SCATS are examples in use around the world [4, 5]. *Third generation* methods employ sophisticated dynamic traffic models and try to find optimal lengths for all phases, given a fixed phase scheme, e.g., by dynamic programming [4]. The exponential complexity of the problem, however, limits these approaches to local view of a few intersections. Reinforcement learning has also been applied [6], but in a way that uses a value function for each car, which is unrealistic in today's world. Common to most approaches is that they deal with the insufficient state information by maintaining a *model* of the traffic situation, derived from available sensor counts. However, imperfections in the model is a further source of errors and performance may consequently suffer. Our methods avoid modelling.

We focus on the system that motivated this research, the Sydney Coordinated Adaptive Traffic System (SCATS) [5]. It is used in major cities throughout Australasia and North America. It provides pre-specified plans that are computed from historical data. There is an additional layer of online adaption based on a *saturation-balancing* algorithm, i.e., SCATS calculates phase timings so that all phases are utilised by vehicles for a fixed percentage of the phase time. Small incremental updates are performed once per cycle. Due to this slow update, rapidly fluctuating traffic conditions pose a problem to SCATS. Furthermore, it does not optimise the global network performance, and its base plans involve hand-tuning to encode offset values.

## 3 Partially Observable MDP Formulation

We cast the traffic problem as a partially observable Markov decision process (POMDP) with states $s$ in a continuous space $\mathbb{S}$. The system is controlled by stochastic actions $\mathbf{a}_t \in \mathbb{A}$ drawn from a random variable (RV) conditioned on the current policy parameters $\theta_t$, and an observation of the current state $\mathbf{o}(s_t)$, according to $\Pr(a_t | \mathbf{o}(s_t), \theta_t)$. In general POMDPs the observation function $\mathbf{o}(s_t)$ is stochastic. NAC, which has only been presented in literature for the fully observable case so far, can be extended to POMDPs given deterministic observations, as this ensures compatibility of the function approximation and avoids injecting noise into the least squares solution. Our traffic POMDP fulfils this requirement. To simplify notation we set $\mathbf{o}_t := \mathbf{o}(s_t)$. The state is an RV evolving as a function of the previous state and action according to $\Pr(s_{t+1} | s_t, \mathbf{a}_t)$. In the case of road traffic these distributions are continuous and complex, making even approximate methods for model based planning in POMDPs difficult to apply. The loop sensors only count cars that pass over them, or are stationary on them. The controller needs access to the history to attempt to resolve some hidden state. We later describe observations constructed from the sensor readings that incorporate some important elements of history for intersection control.

### 3.1 Local Rewards

A common objective function in traffic control is the average vehicle travel time. However, it is impossible to identify a particular car in the system, let alone what its travel time was. Sophisticated

modelling approaches are needed to estimate these quantities. We prefer a direct approach that has the benefits of being trivial to measure from existing sensors and easing the temporal credit assignment problem. For the majority of our experiments we treat each intersection as a local MDP. It is trivial to count all cars that enter the intersection with loop detectors. Therefore, we chose the instant reward $r_{t,i}$ to be the number of cars that entered intersection $i$ over the last time step. The objective for each intersection $i$ is to maximise the normalised *discounted* throughput:

$$R_i(\theta) = \mathbb{E}\left\{ (1-\gamma) \sum\nolimits_{t=0}^{\infty} \gamma^t r_{t,i} \,\middle|\, \theta \right\}.$$

Discounting is important because it ensures that the controller prefers to pass cars through as *early* as possible. While suboptimal policies (in terms of travel time) may achieve the optimal *average* throughput over a time window, the discounted throughput criterion effectively minimises the total waiting time at an intersection in the finite-horizon case [7]. Ignoring effects such as road saturation and driver adaption (which we explore in our experiments), this would result in minimisation of the system wide travel time. The use of local rewards speeds up learning, especially as the number of intersections grows. Bagnell and Ng [8] demonstrate that local rewards alter the sample complexity from worst case $\tilde{\Omega}(I)$, where $I$ is the number of intersections, down to $O(\log I)$. Unfortunately, the value of $R_i(\theta)$ depends directly on the local steady state distribution $\Pr(s_i|\theta)$. Thus changes to the policy of neighbouring intersections can adversely impact intersection $i$, by influencing the distribution of $s_i$. A sufficiently small learning rate allows controllers to adapt to this effectively non-stationary component of the local MDP. We may fail to find the globally optimal cooperative policy without some communication of rewards, but it has proven very effective empirically.

## 4 Natural Actor-Critic Algorithms

Policy-gradient (PG) methods for RL are of interest because it can be easier to learn policies directly than to estimate the exact value of every state of the underlying MDP. While they offer only local convergence guarantees, they do not suffer from the convergence problems exhibited by pure value based methods under function approximation or partial observability [9]. On the other hand, PG methods have suffered from slow convergence compared to value methods due to high variance in the gradient estimates. The natural actor-critic method (NAC) [1] improves this with a combination of PG methods, natural gradients, value estimation, and least-squares temporal-difference Q-learning (LSTD-Q). NAC computes gradient estimates in a batch fashion, followed by a search for the best step size. We introduce an online stochastic gradient ascent using NAC estimates. Stochastic gradient ascent methods often outperform *stochastic* batch methods [2]. We begin with the Bellman equation for fixed parameters $\theta$ where the value of action $a$ in state $s$ is $\mathcal{Q}(s,a)$. This can also be written as the value $\mathcal{V}(s)$ plus the advantage of action $a$ in state $s$, or $\mathcal{A}(s,a)$:

$$\mathcal{Q}(s,a) = \mathcal{V}(s) + \mathcal{A}(s,a) = r(s,a) + \gamma \int_{\mathbb{S}} \Pr(s'|s,a)\mathcal{V}(s')\,ds'. \tag{1}$$

We substitute linear approximators for the value and advantage functions, with parameter vectors $\mathbf{v}$ and $\mathbf{w}$ respectively: $\hat{\mathcal{V}}(s) := \mathbf{o}(s)^{\mathsf{T}}\mathbf{v}$, $\hat{\mathcal{A}}(s,a) := (\nabla_\theta \log \Pr(a|\mathbf{o}(s),\theta))^{\mathsf{T}}\mathbf{w}$, leading to

$$\mathbf{o}(s)^{\mathsf{T}}\mathbf{v} + (\nabla_\theta \log \Pr(a|\mathbf{o}(s),\theta_t))^{\mathsf{T}}\mathbf{w} = r(s,a_t) + \gamma \int_{\mathbb{S}} \Pr(s'|s,a)\mathbf{o}(s')^{\mathsf{T}}\mathbf{v}\,ds'. \tag{2}$$

The surprising choice of $\nabla_\theta \log \Pr(a|\mathbf{o}(s),\theta)$ as features for $\hat{\mathcal{A}}(s,a)$ has the nice property that the parameters $\mathbf{w}$ turn out to be the naturalised gradient of the long-term average reward (and it is *compatible* with the policy parameterisation [9]). To see this we write out the policy-gradient algorithm [9], where $\Pr(s|\theta)$ is the steady state probability of state $s$ and $b(s)$ is a baseline to reduce the variance of gradient estimates

$$\nabla_\theta R(\theta) = \int_{\mathbb{S}} \Pr(s|\theta) \int_{\mathbb{A}} \nabla_\theta \Pr(a|s)(\mathcal{Q}(s,a) - b(s))\,da\,ds. \tag{3}$$

The obvious baseline for making $\mathcal{Q}(s,a)$ zero mean is $b(s) = \mathcal{V}(s)$, which gives $\mathcal{Q}(s,a) - \mathcal{V}(s) = \mathcal{A}(s,a)$. Again, we substitute the linear approximation $\hat{\mathcal{A}}(s,a)$ for $\mathcal{A}(s,a)$ and make use of the fact that our policy is actually a function of $\mathbf{o} := \mathbf{o}(s)$ and $\theta$:

$$\nabla_\theta R(\theta) = \int_{\mathbb{S}} \Pr(s|\theta) \int_{\mathbb{A}} \nabla_\theta \Pr(a|\mathbf{o},\theta)(\nabla_\theta \log \Pr(a|\mathbf{o},\theta))^{\mathsf{T}}\mathbf{w}\,da\,ds.$$

Further substituting $\nabla_\theta \Pr(a|\mathbf{o}, \theta)$ by $\Pr(a|\mathbf{o}, \theta)\nabla_\theta \log \Pr(a|\mathbf{o}, \theta)$ gives

$$\nabla_\theta R(\theta) = \int_{\mathbb{S}} \Pr(s|\theta) \int_{\mathbb{A}} \Pr(a|\mathbf{o}, \theta)\nabla_\theta \log \Pr(a|\mathbf{o}, \theta)(\nabla_\theta \log \Pr(a|\mathbf{o}, \theta))^{\mathsf{T}} \, da \, ds \, \mathbf{w} =: F_\theta \mathbf{w}$$

A key observation is that the matrix $F_\theta$ is the outer product of the log action gradient, integrated over all states and actions. This is exactly the Fisher information matrix [1]. On the other hand, the naturalisation of gradients consists of pre-multiplying the normal gradient by the inverse of the Fisher matrix [10], leading to cancellation of the two Fisher matrices $F_\theta^{-1}\nabla_\theta R(\theta) = \mathbf{w}$.

We return to (2) and reformulate it as a temporal-difference estimate of $\mathcal{Q}(s_t, a_t)$, noting in particular that the integral is replaced by an approximation $\gamma\mathbf{o}_{t+1}\mathbf{v}_t$ of the discounted value of the observed next state. This approximation introduces a zero-mean error $\sigma$. Rewriting as a linear system yields

$$(\mathbf{o}_t - \gamma\mathbf{o}_{t+1})^{\mathsf{T}}\mathbf{v}_t + \sigma(\mathbf{o}_t, s_t, s_{t+1}) + (\nabla_\theta \log \Pr(a_t|\mathbf{o}_t, \theta_t))^{\mathsf{T}}\mathbf{w}_t = r(s_t, a_t).$$

$$[(\nabla_\theta \log \Pr(a_t|\mathbf{o}_t, \theta_t))^{\mathsf{T}}, (\mathbf{o}_t - \gamma\mathbf{o}_{t+1})^{\mathsf{T}}][\mathbf{w}_t^{\mathsf{T}}, \mathbf{v}_t^{\mathsf{T}}]^{\mathsf{T}} + \sigma(\mathbf{o}_t, s_t, s_{t+1}) = r(s_t, a_t)$$

$$z_t[(\nabla_\theta \log \Pr(a_t|\mathbf{o}_t, \theta_t))^{\mathsf{T}}, (\mathbf{o}_t - \gamma\mathbf{o}_{t+1})^{\mathsf{T}}][\mathbf{w}_t^{\mathsf{T}}, \mathbf{v}_t^{\mathsf{T}}]^{\mathsf{T}} + \sigma(\mathbf{o}_t, s_t, s_{t+1}) = z_t r(s_t, a_t) =: \mathbf{g}_t \quad (4)$$

In (4) we pre-multiply both sides by an eligibility trace $z_t$ [11], which gives us the LSTD-Q algorithm for a single sample at step $t$. The NAC algorithm approximates $\mathbf{w}$ by averaging both sides over many time steps $T$ and solving $A_T[\mathbf{w}^{\mathsf{T}}, \mathbf{v}^{\mathsf{T}}]^{\mathsf{T}} = \mathbf{b}_T$, where $A_T = 1/T \sum_{t=1}^{T} z_t[(\nabla_\theta \log \Pr(a_t|\mathbf{o}_t, \theta_t))^{\mathsf{T}}, (\mathbf{o}_t - \gamma\mathbf{o}_{t+1})^{\mathsf{T}}]$ (averaging out the zero-mean noise), and $\mathbf{b}_T = 1/T \sum_{t=0}^{T} \mathbf{g}_t$. By analogy with other second-order gradient methods we can view $A$ as containing curvature information about the optimisation manifold, accelerating learning. In the case of NAC it is an elegant combination of the Fisher information matrix and critic information.

## 4.1 Online Infinite-Horizon Natural Actor Critic

We cannot perform a line search on a real world traffic system because *during* the line search we may try arbitrarily poor step size values. Furthermore, the gradient estimates are noisy, disadvantaging batch methods [2]. E.g., if $\mathbf{w}_t$ is not accurate a line search can step a long way toward a sub-optimal policy and get stuck because the soft-max function used to generate action distributions has a minima for all large parameter values. Methods for preventing line search from going too far typically counteract the advantages of using one at all. Thus, we propose the online version of NAC in Algorithm 1, making a small parameter update at every time step which potentially accelerates convergence because the policy can improve at every step. The main difference between the batch and online versions is the avoidance of the $O(d^3)$ matrix inversion (although Cholesky factorisation can help) for solving $A_T[\mathbf{w}^{\mathsf{T}}, \mathbf{v}^{\mathsf{T}}]^{\mathsf{T}} = \mathbf{b}_T$, where $d = |\theta| + |\mathbf{o}|$. Instead, lines 12 to 15 implement a trick used for Kalman filters: the Sherman-Morrison update of a matrix inverse [2]:

$$(A + \mathbf{z}\mathbf{y}^{\mathsf{T}})^{-1} = A^{-1} - \frac{A^{-1}\mathbf{z}\mathbf{y}^{\mathsf{T}}A^{-1}}{1 + \mathbf{y}^{\mathsf{T}}A^{-1}\mathbf{z}}.$$

In other words, we always work in the inverse space. The update is $O(d^2)$, which is still expensive compared to vanilla PG approaches. Faster methods would be possible if the rank one update of $A$ were of the restricted form $A + \mathbf{z}\mathbf{z}^{\mathsf{T}}$. NAC makes up for expensive computations by requiring orders of magnitude fewer steps to converge to a good policy. We retain the aggregation of $A$,[1] using a rolling average implemented by the $\alpha$ weighting (line 12); however we only use instantaneous estimates $\mathbf{g}_t$ to avoid multiple parameter updates based on the same rewards.

The OLPOMDP, or vanilla, algorithm [12] produces per step gradient estimates from the discounted sum of $\nabla_\theta \log \Pr(a_t|\mathbf{o}_t, \theta_t)$, multiplied by the instant reward $r_t$. This is exactly $\mathbf{w}_t$ if we set $A_t := I$ for all $t$. Other PG approaches [9, 10] are also specialisations of NAC. As the simplest and fastest infinite-horizon algorithm we used OLPOMDP for comparison.

## 5 Policy-Gradient for Traffic Control

PG methods are particularly appealing for traffic control for several reasons. The local search, Monte-Carlo gradient estimates, and local rewards improve scalability. The (almost) direct mapping

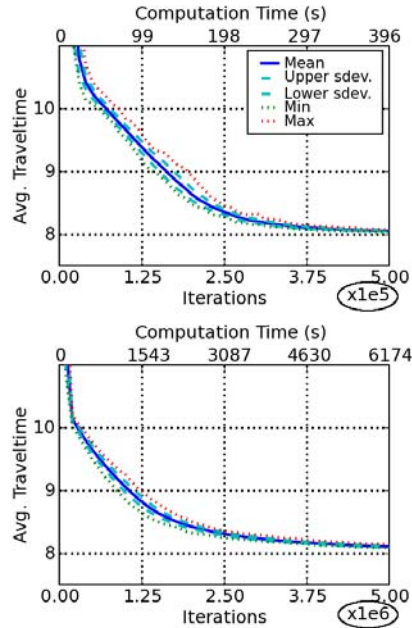

Fig. 2: Convergence properties of NAC (top) compared to OLPOMDP (bottom) over 30 runs in the Offset scenario.

**Alg. 1: An Online Natural Actor-Critic**

1: $t = 1$, $A_1^{-1} = I$, $\theta_1 = [0]$, $\mathbf{z}_1 = [0]$
2: $\epsilon$=step size, $\gamma$=Critic discount, $\lambda$=Actor discount
3: Get observation $\mathbf{o}_1$
4: **while** not converged **do**
5:     Sample action $a_t \sim \Pr(\cdot|\mathbf{o}_t, \theta_t)$
6:     $\mathbf{z}_t = \lambda\mathbf{z}_{t-1} + [\nabla_\theta \log \Pr(a_t|\mathbf{o}_t, \theta_t)^\intercal, \mathbf{o}_t^\intercal]^\intercal$
7:     Do action $a_t$
8:     Get reward $r_t$
9:     $\mathbf{g}_t = r_t\mathbf{z}_t$
10:     Get observation $\mathbf{o}_{t+1}$
11:     $\mathbf{y}_t = [\nabla_\theta \log \Pr(\cdot|\mathbf{o}_t, \theta_t)^\intercal, \mathbf{o}_t^\intercal]^\intercal - \gamma[0^\intercal, \mathbf{o}_{t+1}^\intercal]^\intercal$
12:     $\alpha_t = 1 - \frac{1}{t}$
13:     $\mathbf{u}_t = (1 - \alpha_t)A_{t-1}^{-1}\mathbf{z}_t$
14:     $\mathbf{q}_t^\intercal = \mathbf{y}_t^\intercal A_{t-1}^{-1}$
15:     $A_t^{-1} = \frac{1}{\alpha_t}A_{t-1}^{-1} - \frac{\mathbf{u}_t\mathbf{q}_t^\intercal}{1.0+\mathbf{q}_t^\intercal\mathbf{z}_t}$
16:     $[\mathbf{w}_t^\intercal, \mathbf{v}_t^\intercal]^\intercal = A_t^{-1}\mathbf{g}_t$
17:     $\theta_{t+1} = \theta_t + \epsilon\mathbf{w}_t$
18:     $t \leftarrow t + 1$
19: **end while**

of raw sensor information to controls means that we avoid modelling, and also creates a controller that can react immediately to fluctuations in traffic density. We emphasise that, while SCATS has to adapt a single behaviour slowly, our policy is a rich mapping of sensor data to many different behaviours. Neighbouring intersection controllers cooperate through common observations.

## 5.1 Our Simulator

We implemented a simple traffic simulation system, aiming at maximum simulation speed rather than at an accurate model of traffic flow. We did, however, implement a realistic control interface. We modelled the phase control protocol and sensor system based on information from the Sydney traffic authority. Given that the learning algorithm does not depend directly on a model of the system, just the ability to interact with it, our controller can be plugged into a more accurate simulation without modification. Our simplifying assumptions include: all vehicles move at uniform speed; road length is a multiple of the distance cars travel in one step; we ignore interactions between cars or the relative positions of cars within one road segment except in intersection queues.

Roads are placed in a sparse grid, and intersections may be defined at grid nodes. Every intersection has two queues per incoming road: one queue for right turns and one for going straight or left. Vehicles drive on the left. Every vehicle has a destination intersection that it navigates to via a shortest path. If a driver can select either intersection queue without changing route distance, they choose the one that is currently green, or has the fewest cars in the queue. This models the adaption of drivers to control policies. To account for the gap between a phase ending and the next starting (inter-green time), and the fact that cars start up slowly, we restrict the number of cars that pass through an intersection in the first green step. This factor deters strategies that flip rapidly between phases. We represent saturated traffic conditions through a road capacity parameter. Cars are only allowed to move into the next road segment if the number of cars there does not exceed 20.

## 5.2 The Control Architecture

Commonly intersections have 2 to 6 phases. Ours have 4 phases: for traffic coming from east or west (EW) straight and left turns, EW right turns, north/south (NS) straight and left turns, and NS right turns. At each time step (corresponding to about 5 seconds real-time) the controller decides which

phase to activate in the next step. We do not restrict the order of phases, but to ensure a reasonable policy we enforce the constraint that all phases must be activated at least once within 16 time steps.

The controller input for intersection $i$ is $\mathbf{o}_{t,i}$, constructed as follows: **Cycle duration**: 16 bits, where the $n$th bit is on in the $n$th step of the cycle, supporting time based decisions like offsets. **Current phase**: 4 bits, indicating the previous phase. **Current phase duration**: 5 bits, indicating that we have spent no more than 1, 2, 4, 8 or 13 continuous time steps in the current phase. **Phase durations**: 5 bits per phase, in the same format as current duration, counting the total time spent in each phase in the current cycle. **Detector active**: 8 bits for the 8 loop sensors indicating whether a car is waiting. **Detector history**: 3 bits per loop sensor, indicating a saturation level of more than 0, more than half capacity, or capacity, in the current cycle. **Neighbour information**: 2 bits, giving a delayed comparison of the flows from neighbouring intersections, indicating where traffic is expected from.

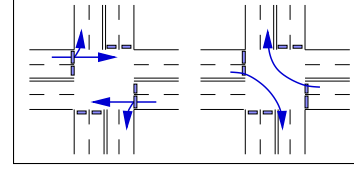

**Fig. 3:** The intersection model, showing 2 phases and detectors.

The controller maps observations $\mathbf{o}_{t,i}$ for intersection $i$ to a probability *distribution* $\tilde{\mathbf{a}}_{t,i}$ over the $P$ phases using a linear approximator with outputs $\mathbf{x}_{t,i}$ and the soft-max function. Let $\theta_i$ be the $P \times |\mathbf{o}_{t,i}|$ matrix of parameters for intersection $i$. We additionally define $\mathbb{U}(p)$ as the unit vector with a 1 in row $p$. Thus, assuming $\exp(\mathbf{x}_{t,i})$ is element-wise exponentiation

$$\mathbf{x}_{t,i} = \theta_i \mathbf{o}_{t,i}\,, \qquad \tilde{\mathbf{a}}_{t,i} = \frac{\exp(\mathbf{x}_{t,i})}{\sum_{p=1}^{P} \exp(\mathbf{x}_{t,i}(p))}\,, \qquad \Pr(a_{t,i} = p|\mathbf{o}_{t,i}, \theta_i) = \tilde{\mathbf{a}}_{t,i}(p);$$

$$\nabla_{\theta_i} \log \Pr(a_{t,i} = p|\mathbf{o}_{t,i}, \theta_i) = (\mathbb{U}(p) - \tilde{\mathbf{a}}_{t,i})\mathbf{o}_{t,i}^{\mathsf{T}}\,, \qquad \nabla_{\theta} \log \Pr(\mathbf{a}_t|\mathbf{o}_t, \theta) = [\nabla_{\theta_1}^{\mathsf{T}}, \ldots, \nabla_{\theta_i}^{\mathsf{T}}]^{\mathsf{T}}.$$

We implemented two baseline controllers for comparison: (1) a uniform controller giving equal length to all phases; (2) A SCATS inspired adaptive controller called SAT that tries to achieve a saturation of 90% (thought to be used by SCATS) for all phases. The exact details of SCATS are not available. We aimed to recreate just the *adaptive* parts of SCATS. It updates the policy once per cycle depending on the current flows [7]. It does not implement the hand-tuned elements of the SCATS controller, such as offsets between neighbouring intersections.

## 6 Experiments

Our first 4 experiments demonstrate scenarios where we expect PG learning to outperform the adaptive SAT controller. The 5th experiment is a large scale experiment where we had no particular prior reason to expect PG to outperform SAT.

**Fluctuating.** This scenario focuses on an intersection in the centre of a crossroads. The traffic volume entering the system on the NS and EW traffic axes is proportional to a sine and cosine function of the time, respectively. Thus the optimal policy at the centre intersection also oscillates with the traffic volume. This scenario is realistic because upstream intersections release periodic bursts of traffic, which then disperse as they travel along the road. SCATS is known to adapt too slowly to deal well with such situations. The road length is 3 units, and vehicles travel through 3 intersections and along 4 roads, leading to an optimal travel time of 12. On average 3 cars enter the system per time step from each direction.

Our results quote the average travel time (TT). Tab. 1 shows that NAC and OLPOMDP both improve upon the uniform controller and SAT. The two PG algorithms get similar results across all scenarios. However, Tab. 2 shows that NAC does so in up to 3 orders of magnitude fewer learning steps, but sometimes requires more CPU time. In a real deployment NAC would be able to keep up with real-time, thus we are much more concerned about reducing learning steps. The tables quote a single run with tuned parameters. To check the reliability of convergence and compare the properties of the two algorithms, Fig. 2 displays the results of 30 runs for both algorithms in one of our scenarios. We also ran the original NAC algorithm, using batch estimates of the direction in a line search to test whether an online NAC is advantageous. We were able to construct a line search that converged faster than online NAC, but always to a significantly worse policy (TT 23 instead of 14.3); or a line search that reached the same policy, but no faster than online. We analysed the sensitivity of

training to the removal of observation features. Performance was degraded after removing any set of observations. Removing multiple observations caused a smooth degradation of the policy [7].

**Burst.** Intersection controllers can learn to "cooperate" by using common observations. In this scenario, we make use of *only* the neighbours feature in the observations, so the controller must use the detector counts of its neighbours to anticipate traffic. The road network is the same as in the Fluctuating scenario. A steady stream of 1 car per step travels EW, so that it is usually good for the centre controller to give maximum time to the EW-straight phase. With the small probability of 0.02 per step, we input a group of 15 cars from the north, travelling south. When this happens, the controller should interrupt its normal policy and give more time to the NS-straight phase.

Table 1 shows that both algorithms learn a good policy in terms of travel time. When examining the learned policies, we noted that the centre intersection controller had indeed learned to switch to the NS-straight phase just in time *before* the cars arrive, something that SAT and SCATS cannot do.

**Offset.** Many drivers have been frustrated by driving along a main street, to be constantly interrupted by red lights. This scenario demonstrates learning an offset between neighbouring intersections, a feature that needs to be hand-tuned in SCATS. We model one arterial with 3 controlled intersections, neglecting any traffic flowing in from side roads. The road length is two units for 4 roads, resulting in an optimal travel time of 8. We restricted the observations to the *cycle duration*, meaning that our controllers learned from time information only.

NAC learned an optimal policy in this scenario. SAT performed badly because it had no means of implementing an offset. We discovered, however, that learning an *optimal* policy is difficult, e.g. we failed for a road length of 3 (given limited time). Learning is hard because intersection $n + 1$ can only begin to learn when intersection $n$ has already converged to an optimal policy.

**Adaptive Driver.** In this scenario, local reward optimisation fails to find the global optimum, so we use the number of cars in the system as the global reward (which minimises the average travel time assuming constant input of cars). This reward is hard to estimate in the real world, but we want to demonstrate the ability of the system to learn cooperative policies using a global reward. Like the previous crossroads scenarios, we have NS and EW streams that interact only at a central intersection (D, in Fig. 4). The streams have the same volume, so the optimal policy splits time uniformly between the NW-straight and EW-straight phases. An additional stream of cars is generated in the south-west corner, at Intersection H, and travels diagonally east to Intersection E. Two equally short routes are available by going straight, or turning east at Intersection F. However, cars that turn east to join the northbound traffic flow must then turn east again at Intersection D, forcing the controller of that intersection to devote time to a third NS-right phase and forcing the main volume of traffic to pause. The optimal strategy is actually to route all cars north from Intersection F, so they join the main eastbound traffic flow. This scenario relies on a driver model that prefers routes with shorter waiting times at intersections among routes of equal distance. Our observations for this scenario consist only of the *phase durations*, informing the controller how much time it has spent in each phase during the current cycle.

Although the average TT of the PG algorithms was only slightly better than SAT's, their policies were radically different. SAT routed cars equally north and east at the critical intersection F, whilst the PG algorithms routed cars north. In this scenario, a slightly larger volume of of vehicles made SAT cause permanent traffic jams, while the PG algorithms still found the correct policy. In this scenario PG even beat our hand-coded optimal deterministic policy because it used stochasticity to find a "mixed" policy, giving phases a fractional number of steps on average.

**Large Scale Optimisation.** In a $10 \times 10$ intersection network, perhaps modelling a central business district, each node potentially produces two cars at each time step according to a randomly initialised probability between 0 and 0.25. The two destinations for the cars from each source are initialised randomly, but stay fixed during the experiment to generate some consistent patterns within the network. Driver adaption and stochastic route choices also create some realistic variance.

We used local rewards and all observations. OLPOMDP gave an average travel time improvement of 20% over SAT even though this scenario was not tailored for our controller. Such savings in a real city would be more than significant. NAC required around 52,000 iterations to improve on SAT. This would be 3 days of experience in the real world to achieve equivalent performance.

Tab. 1: Comparison of best travel times (TT) for all methods and all scenarios. Evaluation for the PG algorithms was over 100,000 steps, after reaching steady state.

| *Scen.* | Random | Unif. | SAT | NAC | OLPOMDP |
|---|---|---|---|---|---|
| Fluct | 250.0 | 102.0 | 21.5 | 14.3 | 13.4 |
| Burst | 197.0 | 35.0 | 18.4 | 13.4 | 13.5 |
| Offset | 17.9 | 15.0 | 12.0 | 8.0 | 8.0 |
| A.D. | 251.0 | 74.2 | 17.2 | 15.8 | 16.0 |
| 100 int. | 60.5 | 54.7 | 35.1 | 29.8 | 27.9 |

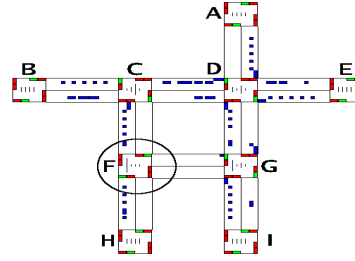

Fig. 4: Adaptive driver scenario.

Tab. 2: Optimisation parameters and run times for all scenarios for the PG algorithms. Optimisation was performed for $t$ steps. 'Secs' is wall-clock time.

| *Scen.* | NAC | | | | | | OLPOMDP | | | | |
|---|---|---|---|---|---|---|---|---|---|---|---|
| | TT | $t$ | secs | $\epsilon$ | $\lambda$ | $\gamma$ | TT | $t$ | secs | $\epsilon$ | $\lambda$ |
| Fluct. | 14.3 | $4.5 \cdot 10^6$ | 860,549 | $10^{-5}$ | 0.9 | 0.95 | 13.4 | $1.1 \cdot 10^9$ | 491,298 | $10^{-3}$ | 0.9 |
| Burst | 13.4 | $4.4 \cdot 10^6$ | 25,454 | $10^{-4}$ | 0.9 | 0.95 | 13.5 | $9.7 \cdot 10^8$ | 35,572 | $10^{-4}$ | 0.9 |
| Offset | 8.0 | $2.1 \cdot 10^6$ | 1,973 | $5 \cdot 10^{-5}$ | 0.98 | 0.9 | 8.0 | $6.3 \cdot 10^8$ | 8,546 | $5 \cdot 10^{-6}$ | 0.98 |
| A.D. | 15.8 | $9.3 \cdot 10^7$ | 867,267 | $10^{-7}$ | 0.98 | 0.95 | 16.0 | $2.2 \cdot 10^9$ | 807,496 | $10^{-6}$ | 0.98 |
| 100 int. | 29.8 | $2.9 \cdot 10^5$ | 1,077,151 | $10^{-4}$ | 0.9 | 0.95 | 27.9 | $3.0 \cdot 10^8$ | 1,029,428 | $10^{-5}$ | 0.9 |

# 7    Conclusion

We described an online stochastic ascent policy-gradient procedure based on the natural actor-critic algorithm. We used it in a distributed road traffic problem to demonstrate where machine learning could improve upon existing proprietary traffic controllers. Our future work will apply this approach to realistic and approved simulators. Improved algorithms will be developed to cope with the increased noise and temporal credit assignment problem inherent in realistic systems.

**Acknowledgments**

National ICT Australia is funded by the Australian Government's Backing Australia's Ability program and the Centre of Excellence program.

**References**

[1] J. Peters, S. Vijayakumar, and S. Schaal. Natural actor-critic. In *Proc. ECML.*, pages 280–291, 2005.

[2] L. Bottou and Y. Le Cun. Large scale online learning. In *Proc. NIPS'2003*, volume 16, 2004.

[3] N. H. Gartner, C. J. Messer, and E. Ajay K. Rathi. *Traffic Flow Theory: A State of the Art Report - Revised Monograph on Traffic Flow Theory*. U.S. Department of Transportation, Transportation Research Board, Washington, D.C., 1992.

[4] M. Papageorgiou. *Traffic Control. In Handbook of Transportation Science*. R. W. Hall, Editor, Kluwer Academic Publishers, Boston, 1999.

[5] A. G. Sims and K. W. Dobinson. The Sydney coordinated adaptive traffic (SCAT) system philosophy and benefits. *IEEE Transactions on Vehicular Technology*, VT-29(2):130–137, 1980.

[6] M. Wiering. Multi-agent reinforcement learning for traffic light control. In *Proc. ICML 2000*, 2000.

[7] S. Richter. Learning traffic control - towards practical traffic control using policy gradients. Diplomarbeit, Albert-Ludwigs-Universität Freiburg, 2006.

[8] J. A. Bagnell and A. Y. Ng. On local rewards and scaling distributed reinforcement learning. In *Proc. NIPS'2005*, volume 18, 2006.

[9] R. S. Sutton, D. McAllester, S. Singh, and Y. Mansour. Policy gradient methods for reinforcement learning with function approximation. In *Proc. NIPS*, volume 12. MIT Press, 2000.

[10] S. Kakade. A natural policy gradient. In *Proc. NIPS'2001*, volume 14, 2002.

[11] J. A. Boyan. Least-squares temporal difference learning. In *Proc. ICML 16*, pages 49–56, 1999.

[12] J. Baxter, P. Bartlett, and L. Weaver. Experiments with infinite-horizon, policy-gradient estimation. *JAIR*, 15:351–381, 2001.

## Footnotes

[1]This means that $A_t$ is a mixture of the Fisher matrices for many parameter values. This is unappealing and we expected a discounted average to yield an $A_t$ that better represents $\theta_t$. However, this performed poorly, perhaps because decaying $\alpha$ mitigates ill-conditioning in the Fisher matrix as parameter values grow [10].
